# Modeling Human Motion
# Using Binary Latent Variables

**Graham W. Taylor, Geoffrey E. Hinton and Sam Roweis**
Dept. of Computer Science
University of Toronto
Toronto, M5S 2Z9 Canada
{gwtaylor,hinton,roweis}@cs.toronto.edu

## Abstract

We propose a non-linear generative model for human motion data that uses an undirected model with binary latent variables and real-valued "visible" variables that represent joint angles. The latent and visible variables at each time step receive directed connections from the visible variables at the last few time-steps. Such an architecture makes on-line inference efficient and allows us to use a simple approximate learning procedure. After training, the model finds a single set of parameters that simultaneously capture several different kinds of motion. We demonstrate the power of our approach by synthesizing various motion sequences and by performing on-line filling in of data lost during motion capture.
**Website**: http://www.cs.toronto.edu/~gwtaylor/publications/nips2006mhmublv/

## 1   Introduction

Recent advances in motion capture technology have fueled interest in the analysis and synthesis of complex human motion for animation and tracking. Models based on the physics of masses and springs have produced some impressive results by using sophisticated "energy-based" learning methods[1] to estimate physical parameters from motion capture data[2]. But if we want to generate realistic human motion, we need to model all the complexities of the real dynamics and this is so difficult to do analytically that learning is likely to be essential. The simplest way to generate new motion sequences based on data is to concatenate parts of training sequences [3]. Another method is to transform motion in the training data to new sequences by learning to adjusting its style or other characteristics[4, 5, 6]. In this paper we focus on model driven analysis and synthesis but avoid the complexities involved in imposing physics-based constraints, relying instead on a "pure" learning approach in which all the knowledge in the model comes from the data.

Data from modern motion capture systems is high-dimensional and contains complex non-linear relationships between the components of the observation vector, which usually represent joint angles with respect to some skeletal structure. Hidden Markov models cannot model such data efficiently because they rely on a single, discrete $K$-state multinomial to represent the history of the time series. To model $N$ bits of information about the past history they require $2^N$ hidden states. To avoid this exponential explosion, we need a model with distributed (i.e. componential) hidden state that has a representational capacity which is linear in the number of components. Linear dynamical systems satisfy this requirement, but they cannot model the complex non-linear dynamics created by the non-linear properties of muscles, contact forces of the foot on the ground and myriad other factors.

## 2 An energy-based model for vectors of real-values

In general, using distributed binary representations for hidden state in directed models of time series makes inference difficult. If, however, we use a Restricted Boltzmann Machine (RBM) to model the probability distribution of the observation vector at each time frame, the posterior over latent variables factorizes completely, making inference easy. Typically, RBMs use binary logistic units for both the visible data and hidden variables, but in our application the data (comprised of joint angles) is continuous. We thus use a modified RBM in which the "visible units" are linear, real-valued variables that have Gaussian noise[7, 8]. The graphical model has a layer of visible units $\boldsymbol{v}$ and a layer of hidden units $\boldsymbol{h}$; there are undirected connections between layers but no connections within a layer. For any setting of the hidden units, the distribution of each visible unit is defined by a parabolic log likelihood function that makes extreme values very improbable:[1]

$$-\log p(\boldsymbol{v}, \boldsymbol{h}) = \sum_i \frac{(v_i - c_i)^2}{2\sigma_i^2} - \sum_j b_j h_j - \sum_{i,j} \frac{v_i}{\sigma_i} h_j w_{ij} + \text{const}, \tag{1}$$

where $\sigma_i$ is the standard deviation of the Gaussian noise for visible unit $i$. (In practice, we rescale our data to have zero mean and unit variance. We have found that fixing $\sigma_i$ at 1 makes the learning work well even though we would expect a good model to predict the data with much higher precision).

The main advantage of using this undirected, "energy-based" model rather than a directed "belief net" is that inference is very easy because the hidden units become conditionally independent when the states of the visible units are observed. The conditional distributions (assuming $\sigma_i = 1$) are:

$$p(h_j = 1 | \boldsymbol{v}) = f(b_j + \sum_i v_i w_{ij}), \tag{2}$$

$$p(v_i | \boldsymbol{h}) = \mathcal{N}(c_i + \sum_j h_j w_{ij}, 1), \tag{3}$$

where $f(\cdot)$ is the logistic function, $\mathcal{N}(\mu, V)$ is a Gaussian, $b_j$ and $c_i$ are the "biases" of hidden unit $j$ and visible unit $i$ respectively, and $w_{ij}$ is the symmetric weight between them.

Maximum likelihood learning is slow in an RBM but learning still works well if we approximately follow the gradient of another function called the contrastive divergence[9]. The learning rule is:

$$\Delta w_{ij} \propto \langle v_i h_j \rangle_{\text{data}} - \langle v_i h_j \rangle_{\text{recon}}, \tag{4}$$

where the first expectation (over hidden unit activations) is with respect to the data distribution and the second expectation is with respect to the distribution of "reconstructed" data. The reconstructions are generated by starting a Markov chain at the data distribution, updating all the hidden units in parallel by sampling (Eq. 2) and then updating all the visible units in parallel by sampling (Eq. 3). For both expectations, the states of the hidden units are conditional on the states of the visible units, not *vice versa*. The learning rule for the hidden biases is just a simplified version of Eq. 4:

$$\Delta b_j \propto \langle h_j \rangle_{\text{data}} - \langle h_j \rangle_{\text{recon}}. \tag{5}$$

### 2.1 The conditional RBM model

The RBM we have described above models static frames of data, but does not incorporate any temporal information. We can model temporal dependencies by treating the visible variables in the previous time slice as additional fixed inputs [10]. Fortunately, this does not complicate inference. We add two types of directed connections (Figure 2): autoregressive connections from the past $n$ configurations (time steps) of the visible units to the current visible configuration, and connections from the past $m$ visibles to the current hidden configuration. The addition of these directed connections turns the RBM into a conditional RBM (CRBM). In our experiments, we have chosen $n = m = 3$. These are, however, tunable parameters and need not be the same for both types of directed connections. To simplify discussion, we will assume $n = m$ and refer to $n$ as the order of the model.

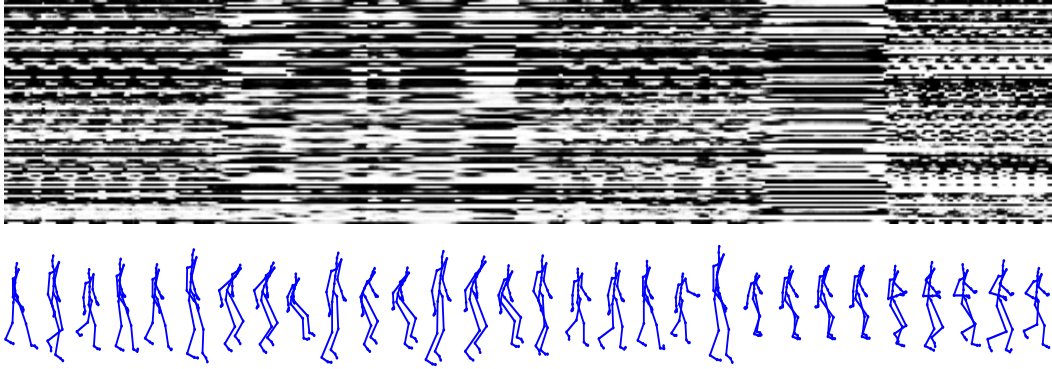

Figure 1: In a trained model, probabilities of each feature being "on" conditional on the data at the visible units. Shown is a 100-hidden unit model, and a sequence which contains (in order) walking, sitting/standing (three times), walking, crouching, and running. Rows represent features, columns represent sequential frames.

Inference in the CRBM is no more difficult than in the standard RBM. Given the data at time $t, t-1, \ldots, t-n$, the hidden units at time $t$ are conditionally independent. We can still use contrastive divergence for training the CRBM. The only change is that when we update the visible and hidden units, we implement the directed connections by treating data from previous time steps as a dynamically changing bias. The contrastive divergence learning rule for hidden biases is given in Eq. 5 and the equivalent learning rule for the temporal connections that determine the dynamically changing hidden unit biases is:

$$\Delta d_{ij}^{(t-q)} \propto v_i^{t-q} \left( \langle h_j^t \rangle_{\text{data}} - \langle h_j^t \rangle_{\text{recon}} \right). \quad (6)$$

where $d_{ij}^{(t-q)}$ is the log-linear parameter (weight) connecting visible unit $i$ at time $t-q$ to hidden unit $j$ for $q = 1..n$. Similarly, the learning rule for the autoregressive connections that determine the dynamically changing visible unit biases is:

$$\Delta a_{ki}^{(t-q)} \propto v_k^{t-q} \left( v_i^t - \langle v_i^t \rangle_{\text{recon}} \right). \quad (7)$$

where $a_{ki}^{(t-q)}$ is the weight from visible unit $k$ at time $t-q$ to visible unit $i$.

The autoregressive weights can model short-term temporal structure very well, leaving the hidden units to model longer-term, higher level structure. During training, the states of the hidden units are determined by both the input they receive from the observed data and the input they receive from the previous time slices. The learning rule for $W$ remains the same as a standard RBM, but has a different effect because the states of the hidden units are now influenced by previous visible units. We do not attempt to model the first $n$ frames of each sequence.

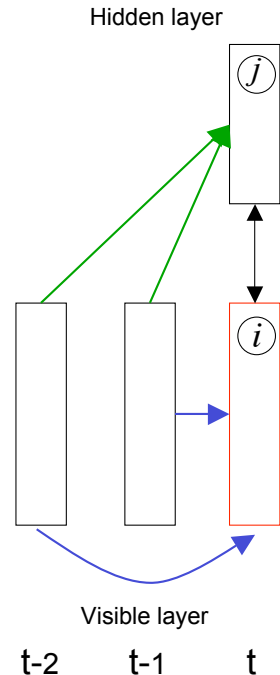

Hidden layer

Visible layer

t-2    t-1    t

Figure 2: Architecture of our model (in our experiments, we use three previous time steps)

While learning a model of motion, we do not need to proceed sequentially through the training data sequences. The updates are only conditional on the past $n$ time steps, not the entire sequence. As long as we isolate "chunks" of frames (the size depending on the order of the directed connections), these can be mixed and formed into mini-batches. To speed up the learning, we assemble these chunks of frames into "balanced" mini-batches of size 100.

We randomly assign chunks to different mini-batches so that the chunks in each mini-batch are as uncorrelated as possible. To save computer memory, time frames are not actually replicated in mini-batches; we simply use indexing to simulate the "chunking" of frames.

## 2.2 Approximations

Our training procedure relies on several approximations, most of which are chosen based on experience training similar networks. While training the CRBM, we replaced $v_i$ in Eq. 4 and Eq. 7 by its expected value and we also used the expected value of $v_i$ when computing the probability of activation of the hidden units. However, to compute the one-step reconstructions of the data, we used stochastically chosen binary values of the hidden units. This prevents the hidden activities from transmitting an unbounded amount of information from the data to the reconstruction [11].

While updating the directed visible-to-hidden connections (Eq. 6) and the symmetric undirected connections (Eq. 4) we used the stochastically chosen binary values of the hidden units in the first term (under the data), but replaced $h_j$ by its expected value in the second term (under the reconstruction). We took this approach because the reconstruction of the data depends on the binary choices made when selecting hidden state. Thus when we infer the hiddens from the reconstructed data, the probabilities are highly correlated with the binary hidden states inferred from the data. On the other hand, we stop after one reconstruction, so the binary choice of hiddens from the reconstruction doesn't correlate with any other terms, and there is no point including this extra noise.

Lastly, we note that the fine-tuning procedure as a whole is making a crude approximation in addition to the one made by contrastive divergence. The inference step, conditional on past visible states, is approximate because it ignores the future (it does not do smoothing). Because of the directed connections, exact inference within the model should include both a forward and backward pass through each sequence (we currently perform only a forward pass). We have avoided a backward pass because missing values create problems in undirected models, so it is hard to perform learning efficiently using the full posterior. Compared with an HMM, the lack of smoothing is a loss, but this is more than offset by the exponential gain in representational power.

# 3 Data gathering and preprocessing

We used data from the CMU Graphics Lab Motion Capture Database as well as from [12] (see acknowledgments). The processed data consists of 3D joint angles derived from 30 (CMU) or 17 (MIT) markers plus a root (coccyx, near the base of the back) orientation and displacement. For both datasets, the original data was captured at 120Hz; we have downsampled it to 30Hz.

Six of the joint angle dimensions in the original CMU data had constant values, so they were eliminated. Each of the remaining joint angles had between one and three degrees of freedom. All of the joint angles and the root orientation were converted from Euler angles to the "exponential map" parameterization [13]. This was done to avoid "gimbal lock" and discontinuities. (The MIT data was already expressed in exponential map form and did not need to be converted.)

We treated the root specially because it encodes a transformation with respect to a fixed global coordinate system. In order to respect physics, we wanted our final representation to be invariant to ground-plane translation and to rotation about the gravitational vertical. We represented each ground-plane translation by an incremental "forwards" vector and an incremental "sideways" vector relative to the direction the person was currently facing, but we represented height non-incrementally by the distance above the ground plane. We represented orientation around the gravitational vertical by the incremental change, but we represented the other two rotational degrees of freedom by the absolute pitch and roll relative to the direction the person was currently facing.

The final dimensionality of our data vectors was 62 (for the CMU data) and 49 (for the MIT data). Note that we eliminated exponential map dimensions that were constant zero (corresponding to joints with a single degree of freedom). As mentioned in Sec. 2, each component of the data was normalized to have zero mean and unit variance.

One advantage of our model is the fact that the data does not need to be heavily preprocessed or dimensionality reduced. Brand and Hertzmann [4] apply PCA to reduce noise and dimensionality. The autoregressive connections in our model can be thought of as doing a kind of "whitening" of the data. Urtasun et al. [6] manually segment data into cycles and sample at regular time intervals using quaternion spherical interpolation. Dimensionality reduction becomes problematic when a wider range of motions is to be modeled.

# 4 Experiments

After training our model using the updates described above, we can demonstrate in several ways what it has learned about the structure of human motion. Perhaps the most direct demonstration, which exploits the fact that it is a probability density model of sequences, is to use the model to generate *de-novo* a number of synthetic motion sequences. Video files of these sequences are available on the website mentioned in the abstract; these motions have not been retouched by hand in any motion editing software. Note that we also do not have to keep a reservoir of training data sequences around for generation - we only need the weights of the model and a few valid frames for initialization.

Causal generation from a learned model can be done on-line with no smoothing, just like the learning procedure. The visible units at the last few time steps determine the effective biases of the visible and hidden units at the current time step. We always keep the previous visible states fixed and perform alternating Gibbs sampling to obtain a joint sample from the conditional RBM. This picks new hidden and visible states that are compatible with each other and with the recent (visible) history. Generation requires initialization with $n$ time steps of the visible units, which implicitly determine the "mode" of motion in which the synthetic sequence will start. We used randomly drawn consecutive frames from the training data as an initial configuration.

## 4.1 Generation of walking and running sequences from a single model

In our first demonstration, we train a single model on data containing both walking and running motions; we then use the learned model to generate both types of motion, depending on how it is initialized. We trained[2] on 23 sequences of walking and 10 sequences of jogging (from subject 35 in the CMU database). After downsampling to 30Hz, the training data consisted of 2813 frames.

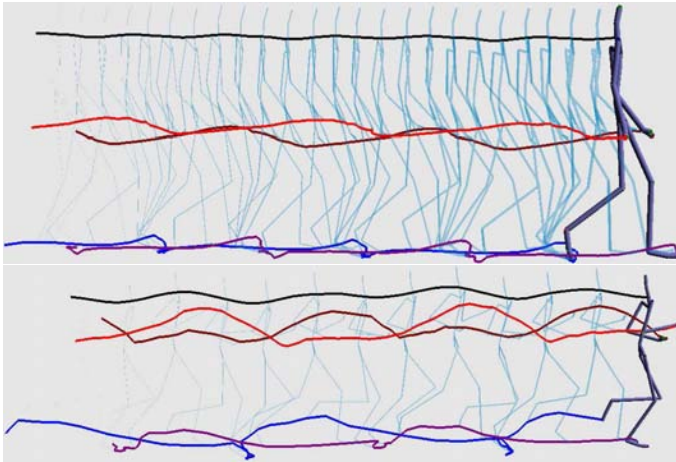

Figure 3: After training, the same model can generate walking (top) and running (bottom) motion (see videos on the website). Each skeleton is 4 frames apart.

Figure 3 shows a walking sequence and a running sequence generated by the same model, using alternating Gibbs sampling (with the probability of hidden units being "on" conditional on the current and previous three visible vectors). Since the training data does not contain any transitions between walking and running (and *vice-versa*), the model will continue to generate walking or running motions depending on where it is initialized.

## 4.2 Learning transitions between various styles

In our second demonstration, we show that our model is capable of learning not only several homogeneous motion styles but also the transitions between them, when the training data itself contains

examples of such transitions. We trained on 9 sequences (from the MIT database, file `Jog1_M`) containing long examples of running and jogging, as well as a few transitions between the two styles. After downsampling to 30Hz, this provided us with 2515 frames. Training was done as before, except that after the model was trained, an identical 200 hidden-unit model was trained on top of the first model (see Sec. 5). The resulting two-level model was used to generate data. A video available on the website demonstrates our model's ability to stochastically transition between various motion styles during a single generated sequence.

### 4.3 Introducing transitions using noise

In our third demonstration, we show how transitions between motion styles can be generated even when such transitions are absent in the data. We use the same model and data as described in Sec. 4.1, where we have learned on separate sequences of walking and running. To generate, we use the same sampling procedure as before, except that at each time we stochastically choose the hidden states (given the current and previous three visible vectors) we add a small amount of Gaussian noise to the hidden state biases. This encourages the model to explore more of the hidden state space without deviating too far the current motion. Applying this "noisy" sampling approach, we see that the generated motion occasionally transitions between learned styles. These transitions appear natural (see the video on the website).

### 4.4 Filling in missing data

Due to the nature of the motion capture process, which can be adversely affected by lighting and environmental effects, as well as noise during recording, motion capture data often contains missing or unusable data. Some markers may disappear ("dropout") for long periods of time due to sensor failure or occlusion. The majority of motion editing software packages contain interpolation methods to fill in missing data, but this leaves the data unnaturally smooth. These methods also rely on the starting and end points of the missing data, so if a marker goes missing until the end of a sequence, naïve interpolation will not work. Such methods often only use the past and future data from the single missing marker to fill in that marker's missing values, but since joint angles are highly correlated, substantial information about the placement of one marker could be gained from the others. Our trained model has the ability to easily fill in such missing data, regardless of where the dropouts occur in a sequence. Due to its approximate inference method which does not rely on a backward pass through the sequence, it also has the ability to fill in such missing data on-line. Filling in missing data with our model is very similar to generation. We simply clamp the known data to the visible units, initialize the missing data to something reasonable (for example, the value at the previous frame), and alternate between stochastically updating the hidden and visible units, with the known visible states held fixed.

To demonstrate filling in, we trained a model exactly as described in Sec. 4.1 except that one walking and one running sequence were left out of the training data to be used as test data. For each of these walking and running test sequences, we erased two different sets of joint angles, starting halfway through the test sequence. These sets were the joints in (1) the left leg, and (2) the entire upper body. As seen in the video files on the website, the quality of the filled-in data is excellent and is hardly distinguishable from the original ground truth of the test sequence. Figure 4 demonstrates the model's ability to predict the three angles of rotation of the left hip.

For the walking sequence (of length 124 frames), we compared our model's performance to nearest neighbor interpolation, a simple method where for each frame, the values on known dimensions are compared to each example in the training set to find the closest match (measured by Euclidean distance in the normalized angle space). The unknown dimensions are then filled in using the matched example. As reconstruction from our model is stochastic, we repeated the experiment 100 times and report the mean. For the missing leg, mean squared reconstruction error per joint using our model was 8.78, measured in normalized joint angle space, and summed over the 62 frames of interest. Using nearest neighbor interpolation, the error was greater: 11.68. For the missing upper body, mean squared reconstruction error per joint using our model was 20.52. Using nearest neighbor interpolation, again the error was greater: 22.20.

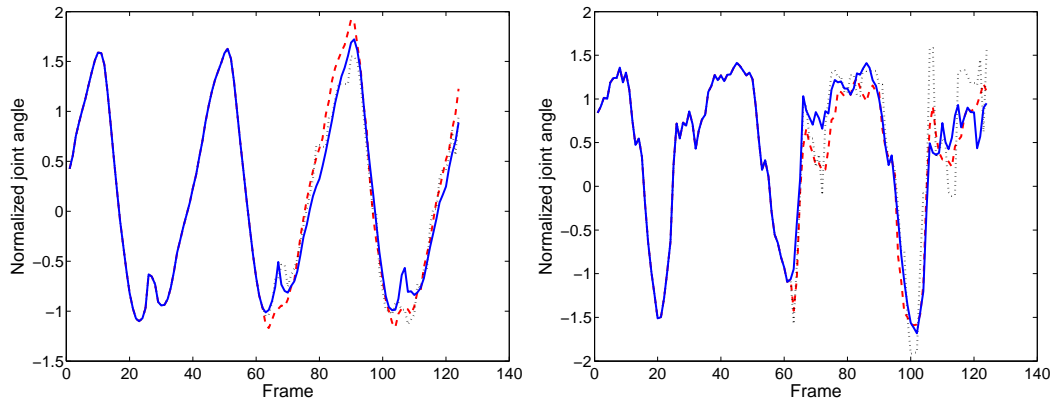

Figure 4: The model successfully fills in missing data using only the previous values of the joint angles (through the temporal connections) and the current angles of other joints (through the RBM connections). Shown are two of the three angles of rotation for the left hip joint (the plot of the third is similar to the first). The original data is shown on a solid line, the model's prediction is shown on a dashed line, and the results of nearest neighbor interpolation are shown on a dotted line (see a video on the website).

## 5  Higher level models

Once we have trained the model, we can add layers like in a Deep Belief Network [14]. The previous layer CRBM is kept, and the sequence of hidden state vectors, while driven by the data, is treated as a new kind of "fully observed" data. The next level CRBM has the same architecture as the first (though we can alter the number of its units) and is trained in the exact same way. Upper levels of the network can then model higher-order structure. This greedy procedure is justified using a variational bound [14]. A two-level model is shown in Figure 5.

We can also consider two special cases of the higher-level model. If we keep only the visible layer, and its $n$-th order directed connections, we have a standard AR($n$) model with Gaussian noise. If we take the two-hidden layer model and delete the first-level autoregressive connections, as well as both sets of visible-to-hidden directed connections, we have a simplified model that can be trained in 2 stages: first learning a static (iid) model of pairs or triples of time frames, then using the inferred hidden states to train a "fully-observed" sigmoid belief net that captures the temporal structure of the hidden states.

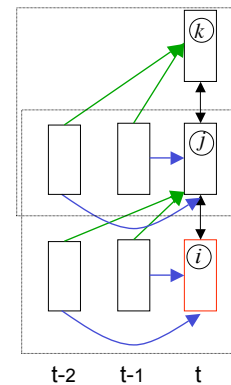

Figure 5: Higher-level models

## 6  Discussion

We have introduced a generative model for human motion based on the idea that local constraints and global dynamics can be learned efficiently by a conditional Restricted Boltzmann Machine. Once trained, our models are able to efficiently capture complex non-linearities in the data without sophisticated pre-processing or dimensionality reduction. The model has been designed with human motion in mind, but should lend itself well to other high-dimensional time series.

In relatively low-dimensional or unstructured data (for example if we were to model a single isolated joint) a single-layer model might be expected to have difficulty since such cyclic time series contain several subsequences which are locally very similar but occur in different phases of the overall cycle. It would be possible to preserve the global phase information by using a much higher order model, but for higher dimensional data such as full body motion capture this is unnecessary because the whole configuration of joint angles and angular velocities never has any phase ambiguity. So the single-layer version of our model actually performs much better on higher-dimensional data.

Models with more hidden layers are able to implicitly model longer-term temporal information, and thus will mitigate this effect.

We have demonstrated that our model can effectively learn different styles of motion, as well as the transitions between these styles. This differentiates our approach from PCA-based approaches which only accurately model cyclic motion, and additionally must build separate models for each type of motion. The ability of the model to transition smoothly, however, is dependent on having sufficient examples of such transitions in the training data. We plan to train on larger datasets encompassing such transitions between various styles of motion. If we augment the data with some static skeletal and identity parameters (in essence mapping a person's unique identity to a set of features), we should be able to use the same generative model for many different people, and generalize individual characteristics from one type of motion to another. Finally, our model is not limited to a single source of data. In the future, we hope to integrate low-level vision data captured at the same time as motion; we could then learn the correlations between the vision stream and the joint angles.

### Acknowledgments

The first data set used in this project was obtained from `mocap.cs.cmu.edu`. This database was created with funding from NSF EIA-0196217. The second data set used in this project was obtained from `http://people.csail.mit.edu/ehsu/work/sig05stf/`. For Matlab playback of motion and generation of videos, we have used Neil Lawrence's motion capture toolbox (`http://www.dcs.shef.ac.uk/~neil/mocap/`).

## Footnotes

[1]For any setting of the parameters, the gradient of the quadratic log likelihood will always overwhelm the gradient due to the weighted input from the binary hidden units provided the value $v_i$ of a visible unit is far enough from its bias, $c_i$.

[2]A 200 hidden-unit CRBM was trained for 4000 passes through the training data, using a third-order model (for directed connections). Weight updates were made after each mini-batch of size 100. The order of the sequences was randomly permuted such that walking and running sequences were distributed throughout the training data.

## References

[1] Y. LeCun, L. Bottou, Y. Bengio, and P. Haffner, "Gradient-based learning applied to document recognition," *Proceedings of the IEEE*, vol. 86, pp. 2278–2324, November 1998.

[2] C. K. Liu, A. Hertzmann, and Z. Popovic, "Learning physics-based motion style with nonlinear inverse optimization," *ACM Trans. Graph.*, vol. 24, no. 3, pp. 1071–1081, 2005.

[3] O. Arikan, D. A. Forsyth, and J. F. O'Brien, "Motion synthesis from annotations," in *Proc. SIGGRAPH*, 2002.

[4] M. Brand and A. Hertzmann, "Style machines.," in *Proc. SIGGRAPH*, pp. 183–192, 2000.

[5] Y. Li, T. Wang, and H.-Y. Shum, "Motion texture: a two-level statistical model for character motion synthesis," in *Proc. SIGGRAPH*, pp. 465–472, 2002.

[6] R. Urtasun, P. Glardon, R. Boulic, D. Thalmann, and P. Fua, "Style-based Motion Synthesis," *Computer Graphics Forum*, vol. 23, no. 4, pp. 1–14, 2004.

[7] M. Welling, M. Rosen-Zvi, and G. E. Hinton, "Exponential family harmoniums with an application to information retrieval.," in *Proc. NIPS 17*, 2005.

[8] Y. Freund and D. Haussler, "Unsupervised learning of distributions of binary vectors using 2-layer networks," in *Proc. NIPS 4*, 1992.

[9] G. E. Hinton, "Training products of experts by minimizing contrastive divergence.," *Neural Comput*, vol. 14, pp. 1771–1800, Aug 2002.

[10] I. Sutskever and G. E. Hinton, "Learning multilevel distributed representations for high-dimensional sequences," Tech. Rep. UTML TR 2006-003, University of Toronto, 2006.

[11] Y. W. Teh and G. E. Hinton, "Rate-coded restricted Boltzmann machines for face recognition," in *Proc. NIPS 13*, 2001.

[12] E. Hsu, K. Pulli, and J. Popoviĉ, "Style translation for human motion," *ACM Trans. Graph.*, vol. 24, no. 3, pp. 1082–1089, 2005.

[13] F. S. Grassia, "Practical parameterization of rotations using the exponential map," *J. Graph. Tools*, vol. 3, no. 3, pp. 29–48, 1998.

[14] G. E. Hinton, S. Osindero, and Y.-W. Teh, "A fast learning algorithm for deep belief nets," *Neural Comp.*, vol. 18, no. 7, pp. 1527–1554, 2006.
